# PROGRAMMABLE SYNAPTIC CHIP FOR ELECTRONIC NEURAL NETWORKS

A. Moopenn, H. Langenbacher, A.P. Thakoor, and S.K. Khanna
Jet Propulsion Laboratory
California Institute of Technology
Pasadena, CA 91009

## ABSTRACT

A binary synaptic matrix chip has been developed for electronic neural networks. The matrix chip contains a programmable 32X32 array of "long channel" NMOSFET binary connection elements implemented in a 3-um bulk CMOS process. Since the neurons are kept off-chip, the synaptic chip serves as a "cascadable" building block for a multi-chip synaptic network as large as 512X512 in size. As an alternative to the programmable NMOSFET (long channel) connection elements, tailored thin film resistors are deposited, in series with FET switches, on some CMOS test chips, to obtain the weak synaptic connections. Although deposition and patterning of the resistors require additional processing steps, they promise substantial savings in silcon area. The performance of a synaptic chip in a 32-neuron breadboard system in an associative memory test application is discussed.

## INTRODUCTION

The highly parallel and distributive architecture of neural networks offers potential advantages in fault-tolerant and high speed associative information processing. For the past few years, there has been a growing interest in developing electronic hardware to investigate the computational capabilities and application potential of neural networks as well as their dynamics and collective properties[1-5]. In an electronic hardware implementation of neural networks[6,7], the neurons (analog processing units) are represented by threshold amplifiers and the synapses linking the neurons by a resistive connection network. The synaptic strengths between neurons (the electrical resistance of the connections) represent the stored information or the computing function of the neural network.

Because of the massive interconectivity of the neurons and the large number of the interconnects required with the increasing number of neurons, implementation of a synaptic network using current LSI/VLSI technology can become very difficult. A synaptic network based on a multi-chip architecture would lessen this difficulty. We have designed, fabricated, and successfully tested CMOS-based programmable synaptic chips which could serve as basic "cascadable" building blocks for a multi-chip electronic neural network. The synaptic chips feature complete programmability of 1024, (32X32) binary synapses. Since the neurons are kept off-chip, the synaptic chips can be connected in parallel, to obtain multiple grey levels of the connection strengths, as well as

"cascaded" to form larger synaptic arrays for an expansion to a 512-neuron system in a feedback or feed-forward architecture. As a research tool, such a system would offer a significant speed improvement over conventional software-based neural network simulations since convergence times for the parallel hardware system would be significantly smaller.

In this paper, we describe the basic design and operation of synaptic CMOS chips incorporating MOSFET's as binary connection elements. The design and fabrication of synaptic test chips with tailored thin film resistors as ballast resistors for controlling power dissipation are also described. Finally, we describe a synaptic chip-based 32-neuron breadboard system in a feedback configuration and discuss its performance in an associative memory test application.

## BINARY SYNAPTIC CMOS CHIP WITH MOSFET CONNECTION ELEMENTS

There are two important design requirements for a binary connection element in a high density synaptic chip. The first requirement is that the connection in the ON state should be "weak" to ensure low overall power dissipation. The required degree of "weakness" of the ON connection largely depends on the synapse density of the chip. If, for example, a synapse density larger than 1000 per chip is desired, a dynamic resistance of the ON connection should be greater than ~100 K-ohms. The second requirement is that to obtain grey scale synapses with up to four bits of precision from binary connections, the consistency of the ON state connection resistance must be better than +/-5 percent, to ensure proper threshold operation of the neurons. Both of the requirements are generally difficult to satisfy simultaneously in conventional VLSI CMOS technology. For example, doped-polysilicon resistors could be used to provide the weak connections, but they are difficult to fabricate with a resistance uniformity of better than 5 percent.

We have used NMOSFET's as connection elements in a multi-chip synaptic network. By designing the NMOSFET's with long channel, both the required high uniformity and high ON state resistance have been obtained. A block diagram of a binary synaptic test chip incorporating NMOSFET's as programmable connection elements is shown in Fig. 1. A photomicrograph of the chip is shown in Fig. 2. The synaptic chip was fabricated through MOSIS (MOS Implementation Service) in a 3-micron, bulk CMOS, two-level metal, P-well technology. The chip contains 1024 synaptic cells arranged in a 32X32 matrix configuration. Each cell consists of a long channel NMOSFET connected in series with another NMOSFET serving as a simple ON/OFF switch. The state of the FET switch is controlled by the output of a latch which can be externally addressed via the ROW/COL address decoders. The 32 analog input lines (from the neuron outputs) and 32 analog output lines (to the neuron inputs) allow a number of such chips to be connected together to form larger connection matrices with up to 4-bit planes.

The long channel NMOSFET can function as either a purely resistive or a constant current source connection element, depending

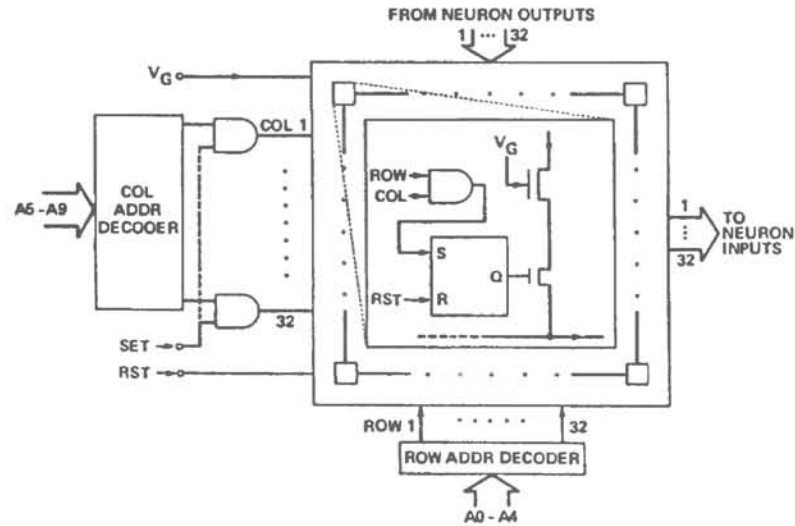

Figure 1.    Block diagram of a 32X32 binary synnaptic chip with long channel NMOSFETs as connection elements.

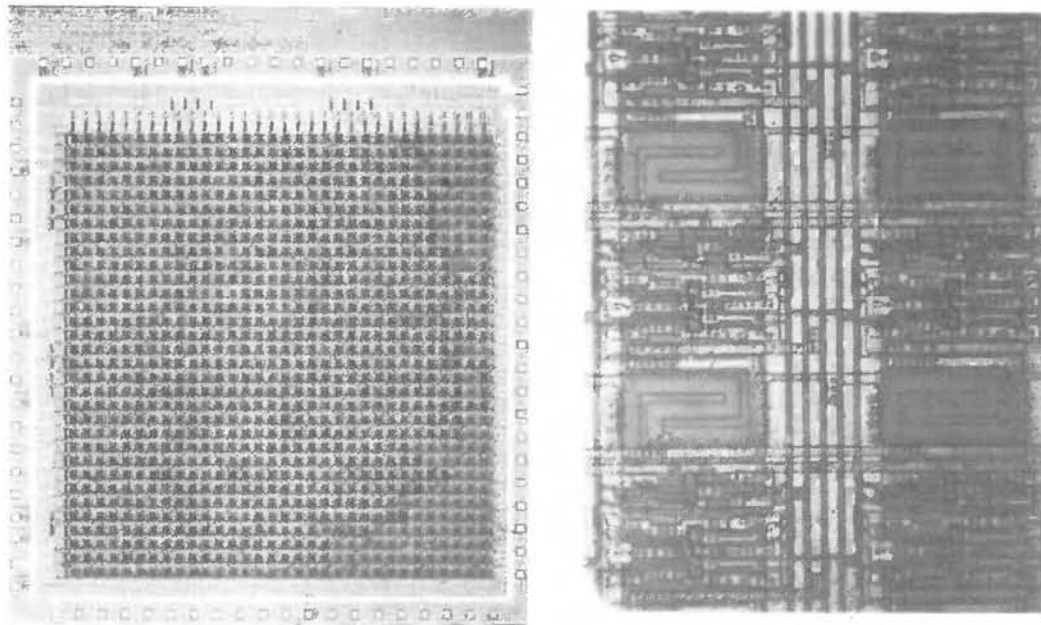

Figure 2.    Photomicrographs of a 32X32 binary  connection CMOS chip. The blowup  on the right shows several synaptic cells; the "S"-shape structures are the long-channel NMOSFETs.

on whether analog or binary output neurons are used.  As a resistive connection, the  NMOSFET's must  operate in the linear region of the transistor's drain I-V characteristics.  In  the linear  region, the channel resistance is approximately given by[8]

$$R_{ON} = (1/K) \ (L/W) \ (V_G - V_{TH})^{-1} .$$

Here, K is a proportionality constant which depends on process parameters, L and W are the channel length and width respectively, $V_G$ is the gate voltage, and $V_{TH}$ is the threshold voltage. The transistor acts as a linear resistor provided the voltage across the channel is much less than the difference of the gate and threshold voltages, and thus dictates the operating voltage range of the connection. The NMOSFET's presently used in our synaptic chip design have a channel length of 244 microns and width of 12 microns. At a gate voltage of 5 volts, a channel resistance of about 200 K-ohms was obtained over an operating voltage range of 1.5 volts. The consistency of the transistor I-V characteristics has been verified to be within +/-3 percent in a single chip and +/-5 percent for chips from different fabrication runs. In the latter case, the transistor characteristics in the linear region can be further matched to within +/-3% by the fine adjustment of their common gate bias.

With two-state neurons, current source connections may be used by operating the transistor in the saturation mode. Provided the voltage across the channel is greater than $(V_G - V_{TH})$, the transistor behaves almost as a constant current source with the saturation current given approximately by[8]

$$I_{ON} = K \ (W/L) \ (V_G - V_{TH})^2 \ .$$

With the appropriate selection of L, W, and $V_G$, it is possible to obtain ON-state currents which vary by two orders of magnitude in values. Figure 3 shows a set of measured I-V curves for a NMOSFET with the channel dimensions, L= 244 microns and W=12 microns and applied gate voltages from 2 to 4.5 volts. To ensure constant current source operation, the neuron's ON-state output should be greater than 3.5 volts. A consistency of the ON-state currents to within +/-5 percent has similarly been observed in a set of chip samples. With current source connections therefore, quantized grey scale synapses with up to 16 grey levels (4 bits) can be realized using a network of binary weighted current sources.

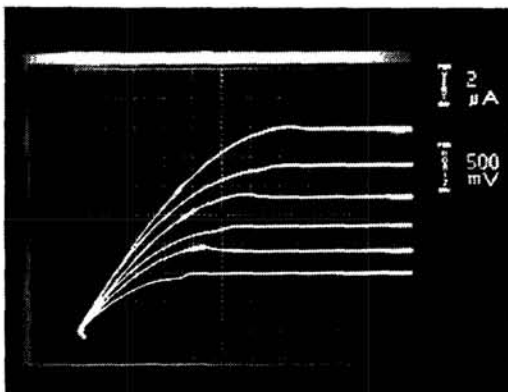

Figure 3. I-V characteristics of an NMOSFET connection element. Channel dimension: L=244 um, W=12um

For proper operation of the NMOSFET connections, the analog output lines (to neuron inputs) should always be held close to ground potential. Moreover, the voltages at the analog input lines must be at or above ground potential. Since the current normally flows from the analog input to the output, the NMOSFET's may be used
as either all excitatory or inhibitory type connections. However,
the complementary connection function can be realized using long
channel PMOSFET's in series with PMOSFET switches. For a PMOSFET
connection, the voltage of an analog input line would be at or below
ground. Furthermore, due to the difference in the mobilites of
electrons and holes in the channel, a PMOSFET used as a resistive
connection has a channel resistance about twice as large as an
NMOSFET with the same channel dimension. This fact results in a
subtantial reduction in the size of PMOSFET needed.

THIN FILM RESISTOR CONNECTIONS

The use of MOSFET's as connection elements in a CMOS synaptic
matrix chip has the major advantage that the complete device can be
readily fabricated in a conventional CMOS production run. However,
the main disadvantages are the large area (required for the long
channel) for the MOSFET's connections and their non-symmetrical
inhibitory/excitatory functional characteristics. The large overall
gate area not only substantially limits the number of synapses that
can be fabricated on a single chip, but the transistors are more
susceptible to processing defects which can lead to excessive gate
leakage and thus reduce chip yield considerably. An alternate
approach is simply to use resistors in place of MOSFET's. We have
investigated one such approach where thin film resistors are
deposited on top of the passivation layer of CMOS-processed chips as
an additional special processing step to the normal CMOS fabrication
run. With an appropriate choice of resistive materials, a dense
array of resistive connections with highly uniform resistance of up
to 10 M-ohms appears feasible.
Several candidate materials, including a cermet based on
platinum/aluminum oxide, and amorphous semiconductor/metal alloys
such as a-Ge:Cu and a-Ge:Al, have been examined for their applica-
bility as thin film resistor connections. These materials are of
particular interest since their resistivity can easily be tailored
in the desired semiconducting range of 1-10 ohm-cm by controlling
the metal content[9]. The a-Ge/metal films are deposited by thermal
evaporation of presynthesized alloys of the desired composition in
high vacuum, whereas platinum/aluminum oxide films are deposited by
co-sputtering from platinum and aluminum oxide targets in a high
purity argon and oxygen gas mixture. Room temperature resistivities
in the 0.1 to 100 ohm-cm range have been obtained by varying the
metal content in these materials. Other factors which would also
determine their suitability include their device processing and
material compatibilities and their stability with time, temperat-
ure, and extended application of normal operating electric current.
The temperature coefficient of resistance (TCR) of these materials
at room temperature has been measured to be in the 2000 to 6000 ppm
range. Because of their relatively high TCR's, the need for weak
connections to reduce the effect of localized heating is especially
important here. The a-Ge/metal alloy films are observed to be
relatively stable with exposure to air for temperatures below 130° C.

The platinum/aluminum oxide film stabilize with time after annealing in air for several hours at 130° C.

Sample test arrays of thin film resistors based on the described materials have been fabricated to test their consistency. The resistors, with a nominal resistance of 1 M-ohm, were deposited on a glass substrate in a 40X40 array over a 0.4cm by 0.4cm area. Variation in the measured resistance in these test arrays has been found to be from +/- 2-5 percent for all three materials. Smaller test arrays of a-Ge:Cu thin film resistors on CMOS test chips have also been fabricated. A photo-micrograph of a CMOS synaptic test chip containing a 4X4 array of a-Ge:Cu thin film resistors is shown in Fig. 4. Windows in the passivation layer of silicon nitride (SiN) were opened in the final processing step of a normal CMOS fabrication run to provide access to the aluminum metal for electrical contacts. A layer of resistive material was deposited and patterned by lift-off. A layer of buffer metal of platinum or nickel was then deposited by RF sputtering and also patterned by lift-off. The buffer metal pads serve as a conducting bridges for connecting the aluminum electrodes to the thin film resistors. In addition to providing a reliable ohmic contact to the aluminum and resistor, it also provides conformal step coverage over the silicon nitride window edge. The resistor elements on the test chip are 100 micron long, 10 micron wide with a thickness of about 1500 angstroms and a nominal resistance of 250 K-ohms. Resistance variations from 10-20 percent have been observed in several such test arrays. The unusually large variation is largely due to the surface roughness of the chip passivation layer. As one possible solution, a thin spin-

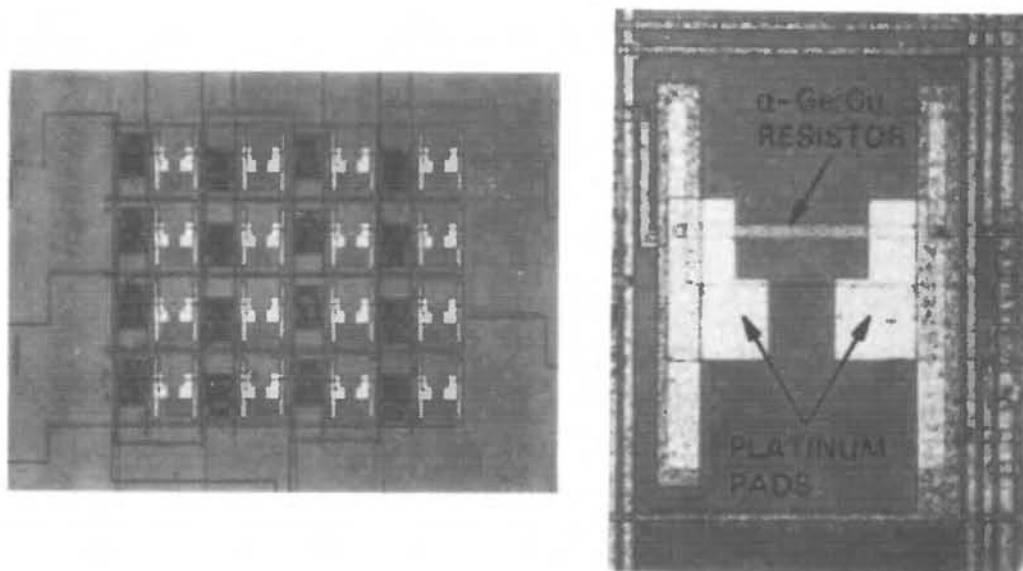

Figure 4. Photomicrographs of a CMOS synaptic test chip with a 4X4 array of a-Ge:Cu thin film resistors. The nominal resistance was 250 K-ohms.

on coating of an insulating material such as polyimide to smooth out the surface of the passivation layer prior to depositing the resistors is under investigation.

## SYNAPTIC CHIP-BASED 32-NEURON BREADBOARD SYSTEM

A 32-neuron breadboard system utilizing an array of discrete neuron electronics has been fabricated to evaluate the operation of 32X32 binary synaptic CMOS chips with NMOSFET connection elements. Each neuron consists of an operational amplifier configured as a current to voltage converter (with virtual ground input) followed by a fixed-gain voltage difference amplifier. The overall time constant of the neurons is approximately 10 microseconds. The neuron array is interfaced directly to the synaptic chip in a full feedback configuration. The system also contains prompt electronics consisting of a programmable array of RC discharging circuits with a relaxation time of approximately 5 microseconds. The prompt hardware allows the neuron states to be initialized by precharging the selected capacitors in the RC circuits. A microcomputer interfaced to the breadboard system is used for programming the synaptic matrix chip, controlling the prompt electronics, and reading the neuron outputs.

The stability of the breadboard system is tested in an associative memory feedback configuration[6]. A dozen random dilute-coded binary vectors are stored using the following simplified outer-product storage scheme:

$$T_{ij} = \begin{cases} -1 & \text{if } \sum_{S} V_i^S V_j^S = 0 \\ 0 & \text{otherwise.} \end{cases}$$

In this scheme, the feedback matrix consists of only inhibitory (-1) or open (0) connections. The neurons are set to be normally ON and are driven OFF when inhibited by another neuron via the feedback matrix. The system exhibits excellent stability and associative recall performance. Convergence to a nearest stored memory in Hamming distance is always observed for any given input cue. Figure 5 shows some typical neuron output traces for a given test prompt and a set of stored memories. The top traces show the response of two neurons that are initially set ON; the bottom traces for two other neurons initially set OFF. Convergence times of 10-50 microseconds have been observed, depending on the prompt conditions, but are primarily governed by the speed of the neurons.

## CONCLUSIONS

Synaptic CMOS chips containing 1024 programmable binary synapses in a 32X32 array have been designed, fabricated, and tested. These synaptic chips are designed to serve as basic building blocks for large multi-chip synaptic networks. The use of long channel MOSFET's as either resistive or current source connection elements meets the "weak" connection and consistency

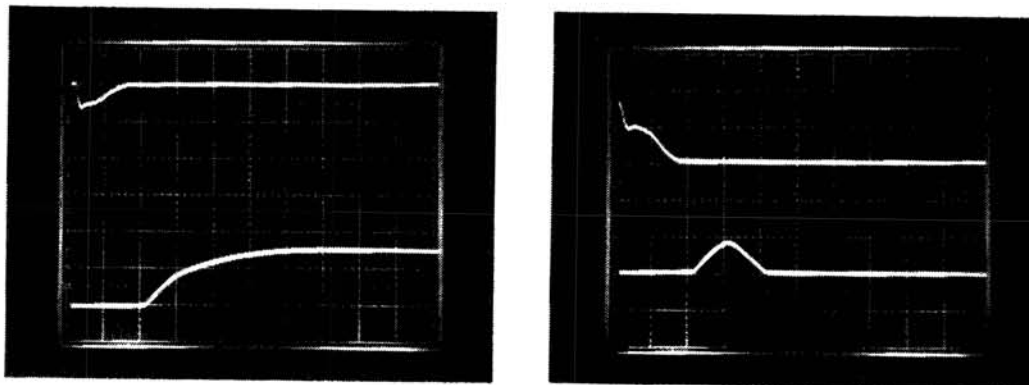

Figure 5. Typical neuron response curves for a test prompt input. (Horiz scale: 10 microseconds per div)

requirements. Alternately, CMOS-based synaptic test chips with specially deposited thin film high-valued resistors, in series with FET switches, offer an attractive approach to high density programmable synaptic chips. A 32-neuron breadboard system incorporating a 32X32 NMOSFET synaptic chip and a feedback configuration exhibits excellent stability and associative recall performance as an associative memory. Using discrete neuron array, convergence times of 10-50 microseconds have been demonstrated. With optimization of the input/output wiring layout and the use of high speed neuron electronics, convergence times can certainly be reduced to less than a microsecond.

## ACKNOWLEDGEMENTS

This work was performed by the Jet Propulsion Laboratory, California Institute of Technology, and was sponsored by the Joint Tactical Fusion Program Office, through an agreement with the National Aeronautics and Space Administration. The authors would like to thank John Lambe for his invaluable suggestions, T. Duong for his assistance in the breadboard hardware development, J. Lamb and S. Thakoor for their help in the thin film resistor deposition, and R. Nixon and S. Chang for their assistance in the chip layout design.

## REFERENCES

1. J. Lambe, A. Moopenn, and A.P. Thakoor, Proc. AIAA/ACM/NASA/-IEEE Computers in Aerospace V, 160 (1985)
2. A.P. Thakoor, J.L. Lamb, A. Moopenn, and S.K. Khanna, MRS Proc. 95, 627 (1987)
3. W. Hubbard, D. Schwartz, J. Denker, H.P. Graf, R. Howard, L. Jackel, B. Straughn, and D. Tennant, AIP Conf. Proc. 151, 227 (1986)
4. M.A. Sivilotti, M.R. Emerling, and C. Mead, AIP Conf. Proc. 151, 408 (1986)
5. J.P. Sage, K. Thompson, and R.S. Withers, AIP Conf. Proc. 151,

381 (1986)

6.  J.J. Hopfield, Proc. Nat. Acad. Sci., 81, 3088 (1984)
7.  J.J. Hopfield, Proc. Nat. Acad. Sci., 79, 2554 (1982)
8.  S.M. Sze, "Semiconductor Devices-Physics and Technology," (Wiley, New York, 1985) p.205
9.  J.L. Lamb, A.P. Thakoor, A. Moopenn, and S.K. Khanna, J. Vac. Sci. Tech., A 5(4), 1407 (1987)
